# Probabilistic Joint Image Segmentation and Labeling[*]

**Adrian Ion**[1,2]**, Joao Carreira**[1]**, Cristian Sminchisescu**[1]
[1]Faculty of Mathematics and Natural Sciences, University of Bonn
[2] PRIP, Vienna University of Technology & Institute of Science and Technology, Austria
`{ion,carreira,cristian.sminchisescu}@ins.uni-bonn.de`

## Abstract

We present a joint image segmentation and labeling model (JSL) which, given a bag of figure-ground segment hypotheses extracted at multiple image locations and scales, constructs a joint probability distribution over *both* the compatible image interpretations (*tilings* or image segmentations) composed from those segments, *and* over their *labeling* into categories. The process of drawing samples from the joint distribution can be interpreted as first sampling tilings, modeled as maximal cliques, from a graph connecting spatially non-overlapping segments in the bag [1], followed by sampling labels for those segments, conditioned on the choice of a particular tiling. We learn the segmentation and labeling parameters jointly, based on Maximum Likelihood with a novel Incremental Saddle Point estimation procedure. The partition function over tilings and labelings is increasingly more accurately approximated by including incorrect configurations that a not-yet-competent model rates probable during learning. We show that the proposed methodology matches the current state of the art in the Stanford dataset [2], as well as in VOC2010, where 41.7% accuracy on the test set is achieved.

## 1 Introduction

One of the main goals of scene understanding is the semantic segmentation of images: label a diverse set of object properties, at multiple scales, while at the same time identifying the spatial extent over which such properties hold. For instance, an image may be segmented into things (man-made objects, people or animals), amorphous regions or stuff like grass or sky, or main geometric properties like the ground plane or the vertical planes corresponding to buildings in the scene. The optimal identification of such properties requires inference over spatial supports of different levels of granularity, and such regions may often overlap. It appears to be now well understood that a successful extraction of such properties requires models that can make inferences over adaptive spatial neighborhoods that span well beyond patches around individual pixels. Incorporating segmentation information to inform the labeling process has recently become an increasingly active research area. While initially inferences were restricted to super-pixel segmentations, recent trends emphasize joint models with capabilities to represent the uncertainty in the segmentation process [2, 4, 5, 6, 7]. One difficulty is the selection of segments that have adequate spatial support for reliable labeling, and a second major difficulty is the design of models where both the segmentation and the labeling layers can be learned jointly. In this paper, we present a joint image segmentation and labeling model (JSL) which, given a bag of possibly overlapping figure-ground (binary) segment hypotheses, extracted independently at multiple image locations and scales, constructs a joint probability distribution over *both* the compatible image interpretations (or tilings) assembled from those segments, *and* over their labels. For learning, we present a procedure based on Maximum Likelihood, where the partition function over tilings and labelings is increasingly more accurately approximated in each iteration, by including incorrect configurations that the model rates probable. This prevents

---

[*]Supported, in part, by the EC, under MCEXT-025481, and by CNCSIS-UEFISCU, PNII-RU-RC-2/2009.

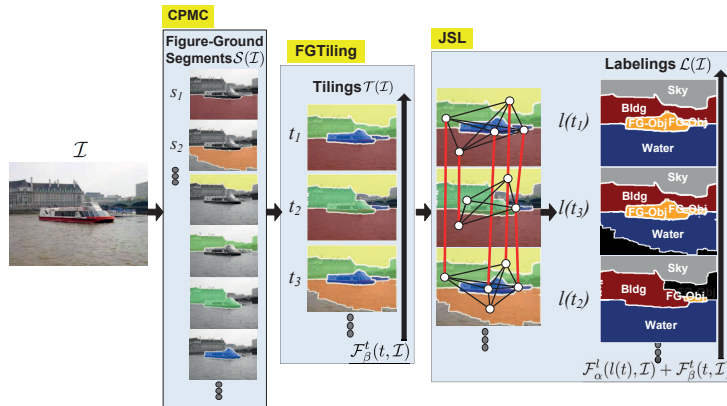

Figure 1: Overview of our joint segment composition and categorization framework. Given an image $\mathcal{I}$, we extract a bag $\mathcal{S}$ of figure-ground segmentations, constrained at different spatial locations and scales, using the CPMC algorithm [3] and retain the *figure* segments (other algorithms can be used for segment bagging). Segments are composed into image interpretations (tilings) by FGTiling [1]. In brief, segments become nodes in a *consistency graph* where any two segments that do not spatially overlap are connected by an edge. Valid compositions (tilings) are obtained by computing maximal cliques in the consistency graph. Multiple tilings are usually generated for each image. Tilings consist of subsets of segments in $\mathcal{S}$, and may induce *residual regions* that contain pixels not belonging to any of the segments selected in a particular tiling. For labeling (JSL), configurations are scored based on both their *category-dependent* properties measured by $\mathcal{F}_\alpha^l$, and by their mid-level *category-independent* properties measured by $\mathcal{F}_\beta^t$ over the *dependency graph*—a subset of the *consistency graph* connecting only spatially neighboring segments that share a boundary. The model parameters $\theta = [\alpha^\top \ \beta^\top]^\top$ are jointly learned using Maximum Likelihood based on a novel incremental Saddle Point partition function approximation. Notice that a segment appearing in different tilings of an image $\mathcal{I}$ is constrained to have the same label (red vertical edges).

cyclic behavior and leads to a stable optimization process. The method *jointly learns* both the mid-level, category-independent parameters of a segment composition model, and the category-sensitive parameters of a labeling model for those segments. To our knowledge this is the first model for joint image segmentation and labeling, that accommodates both inference and learning, within a common, consistent probabilistic framework. We show that our procedure matches the state of the art in the Stanford [2], as well as the VOC2010 dataset, where 41.7% accuracy on the test set is achieved. Our framework is reviewed in fig. 1.

## 1.1 Related Work

One approach to recognize the elements of an image would be to accurately partition it into regions based on low and mid-level statistical regularities, and then label those regions, as pursued by Barnard *et al.* [8]. The labeling problem can then be reduced to a relatively small number of classification problems. However, most existing mid-level segmentation algorithms cannot generate one unique, yet accurate segmentation per image, across multiple images, for the same set of generic parameters [9, 10]. To achieve the best recognition, some tasks might require multiple overlapping spatial supports which can only be provided by different segmentations.

Segmenting object parts or regions can be done at a finer granularity, with labels decided locally, at the level of pixels [11, 12, 13] or superpixels [14, 15], based on measurements collected over neighborhoods with limited spatial support. Inconsistent label configurations can be resolved by smoothing neighboring responses, or by encouraging consistency among the labels belonging to regions with similar low-level properties [16, 13]. The models are effective when local appearance statistics are discriminative, as in the case of amorphous stuff (water, grass), but inference is harder to constrain for shape recognition, which requires longer-range interactions among groups of measurements. One way to introduce constraints is by estimating the categories likely to occur in the image using global classifiers, then bias inference to that label distribution [12, 13, 15].

A complementary research trend is to segment and recognize categories based on features extracted over competing image regions with larger spatial support (extended regions). The extended regions can be rectangles produced by bounding box detectors [17, 2]. The responses are combined in a single pixel or superpixel layer [7, 18, 17, 6] to obtain the final labeling. Extended regions can also arise from multiple full-image segmentations [7, 18, 6]. By computing segmentations multiple times with different parameters, chances increase that some of the segments are accurate. Multiple segmentations can also be aggregated in an inclusion hierarchy [19, 5], instead of being obtained independently. The work of Tu *et al.* [20] uses generative models to drive the sequential re-segmentation process, formulated as Data Driven Markov Chain Monte Carlo inference. Recently, Gould *et al.* [2] proposed a model for segmentation and labeling where new region hypotheses were generated through a sequential procedure, where uniform label swaps for all the pixels contained inside individual segment proposals are accepted if they reduce the value of a global energy function. Kumar and Koller [4] proposed an improved joint inference using dual-decomposition. Our approach for segmentation and labeling is layered rather than simultaneous, and learning for the segmentation and labeling parameters is performed jointly (rather than separately), in a probabilistic framework.

## 2  Probabilistic Segmentation and Labeling

Let $\mathcal{S} = \{s_1, s_2, \dots\}$, be a set (bag) of segments from an image $\mathcal{I}$. In our case, the segments $s_i$ are obtained using the publicly available CPMC algorithm [3], and represent different figure-ground hypotheses, computed independently by applying constraints at various spatial locations and scales in the image.[1] Subsets of segments in the bag $\mathcal{S}$ form the power set $\mathcal{P}(\mathcal{S})$, with $2^{|\mathcal{S}|}$ possible elements. We focus on a restriction of the power set of an image, its *tiling set* $\mathcal{T}(\mathcal{I})$, with the property that all segments contained in any subset (or **tiling**) do not spatially overlap and the subset is maximal: $\mathcal{T}(\mathcal{I}) = \{t = \{\dots s_i, \dots s_j, \dots\} \in \mathcal{P}(\mathcal{S}),\ \text{s.t.}\ \forall i, j,\ \text{overlap}(s_i, s_j) = 0\}$. Each tiling $t$ in $\mathcal{T}(\mathcal{I})$ can have its segments labeled with one of $L$ possible category labels. We call a **labeling** the mapping obtained by assigning labels to segments in a tiling $l(t) = \{l_1, \dots, l_{|t|}\}$, with $l_i \in \{1, \dots, L\}$ the label of segment $s_i$, and $|l(t)| = |t|$ (one label corresponds to one segment).[2] Let $\mathcal{L}(\mathcal{I})$ be the set of all possible labelings for image $\mathcal{I}$ with

$$|\mathcal{L}(\mathcal{I})| = \sum_{t \in \mathcal{T}(\mathcal{I})} L^{|t|} \tag{1}$$

where we sum over all valid segment compositions (tilings) of an image, $\mathcal{T}(\mathcal{I})$, and the label space of each. We define a *joint* probability distribution over tilings and their corresponding labelings,

$$p_\theta(l(t), t, \mathcal{I}) = \frac{1}{Z_\theta(\mathcal{I})} \exp \mathcal{F}_\theta(l(t), t, \mathcal{I}) \tag{2}$$

where $Z_\theta(\mathcal{I}) = \sum_t \sum_{l(t)} \exp \mathcal{F}_\theta(l(t), t, \mathcal{I})$ is the normalizer or partition function, $l(t) \in \mathcal{L}(\mathcal{I}), t \in \mathcal{T}(\mathcal{I})$, and $\theta$ the parameters of the model. It is a constrained probability distribution defined over two sets: a set of segments in a tiling and an index set of labels for those segments, both of the same cardinality. $\mathcal{F}_\theta$ is defined as

$$\mathcal{F}_\theta(l(t), t, \mathcal{I}) = \mathcal{F}_\alpha^l(l(t), \mathcal{I}) + \mathcal{F}_\beta^t(t, \mathcal{I}) \tag{3}$$

with parameters $\theta = [\alpha^\top\ \beta^\top]^\top$. The additive decomposition can be viewed as the sum of one term, $\mathcal{F}_\beta^t(t, \mathcal{I})$, encoding a mid-level, *category independent* score of a particular tiling $t$, and another *category-dependent* score, $\mathcal{F}_\alpha^l(l(t), \mathcal{I})$, encoding the potential of a labeling $l(t)$ for that tiling $t$. The components $\mathcal{F}_\alpha^l(l(t), \mathcal{I})$ and $\mathcal{F}_\beta^t(t, \mathcal{I})$ are defined as interactions over unary and pairwise terms. The potential of a labeling is

$$\mathcal{F}_\alpha^l(l(t), \mathcal{I}) = \sum_{s_i \in t} \Phi_{l_i}^l(s_i, \alpha) + \sum_{s_i \in t} \sum_{s_j \in \mathcal{N}_{s_i}^l} \Psi_{l_i, l_j}^l(s_i, s_j, \alpha) \tag{4}$$

with $\Phi_{l_i}^l$ and $\Psi_{l_i, l_j}^l$ unary and pairwise, label-dependent potentials, and $\mathcal{N}_{s_i}^l$ the label relevant neighborhood of $s_i$. In our experiments we take $\mathcal{N}_{s_i}^l = t \setminus \{s_i\}$. The unary and pairwise terms are linear

in the parameters, e.g. $\Phi^l_{l_i}(s_i, \alpha) = \alpha^\top \Phi^l_{l_i}(s_i)$. For example $\Phi^l_{l_i}(s_i, \alpha)$ encodes how likely it is for segment $s_i$ to exhibit the regularities typical of objects belonging to class $l_i$. The potential of a tiling is defined as

$$\mathcal{F}^t_\beta(t, \mathcal{I}) = \sum_{s_i \in t} \Phi^t(s_i, \beta) + \sum_{s_i \in t} \sum_{s_j \in \mathcal{N}^t_{s_i}} \Psi^t(s_i, s_j, \beta) \tag{5}$$

with $\Phi^t$ and $\Psi^t$ unary and pairwise, label-independent potential functions, and $\mathcal{N}^t_{s_i}$ the local image neighborhood i.e. $\mathcal{N}^t_{s_i} = \{s_j \in t \mid s_i, s_j \text{ share a boundary part and do not overlap}\}$. Both terms $\Phi^t$ and $\Psi^t$ are linear in the parameters, similar to the components of the category dependent potential $\mathcal{F}^l_\alpha(l(t), \mathcal{I})$. For example $\Phi^t(s_i, \alpha)$ encodes how likely is that segment $s_i$ exhibits generic object regularities (details on the segmentation model $\mathcal{F}^t_\beta(t, \mathcal{I})$ can be found in [1]).

**Inference:** Given an image $\mathcal{I}$, inference for the optimal tiling and labeling $(l^*(t^*), t^*)$ is given by

$$(l^*(t^*), t^*) = \underset{l(t), t}{\operatorname{argmax}} p_\theta(l(t), t, \mathcal{I}) \tag{6}$$

Our inference methodology is described in sec. 3.

**Learning:** During learning we optimize the parameters $\theta$ that maximize the likelihood (ML) of the ground truth under our model:

$$\theta^\star = \underset{\theta}{\operatorname{argmax}} \prod_{\mathcal{I}} p_\theta(l^{\mathcal{I}}(t^{\mathcal{I}}), t^{\mathcal{I}}, \mathcal{I}) = \underset{\theta}{\operatorname{argmax}} \sum_{\mathcal{I}} \left[ \mathcal{F}_\theta(l^{\mathcal{I}}(t^{\mathcal{I}}), t^{\mathcal{I}}, \mathcal{I}) - \log Z_\theta(\mathcal{I}) \right] \tag{7}$$

where $(l^{\mathcal{I}}(t^{\mathcal{I}}), t^{\mathcal{I}})$ are ground truth labeled tilings for image $\mathcal{I}$. Our learning methodology, including an incremental saddle point approximation for the partition function is presented in sec. 4.

## 3    Inference for Tilings and Labelings

Given an image where a bag $\mathcal{S}$ of multiple figure-ground segments has been extracted using CPMC [3], inference is performed by first composing a number of plausible tilings from subsets of the segments, then labeling each tiling using spatial inference methods.

The inference algorithm for computing (sampling) tilings associates each segment to a node in a consistency graph where an edge exists between all pairs of nodes corresponding to segments that do not spatially overlap. The cliques of the consistency graph correspond to alternative segmentations of the image constructed from the basic segments. The algorithm described in [1] can efficiently find a number of plausible maximal weighted cliques, scored by (5). A maximum of $|\mathcal{S}|$ distinct maximal cliques (tilings) are returned, and each segment $s_i$ is contained in at least one of them.

Inference for the labels of the segments in each tiling can be performed using any number of reliable methods—in this work we use tree-reweighted belief propagation TRW-S [21]. The maximum in (6) is computed by selecting the labeling with the highest probability (2) among the tilings generated by the segmentation algorithm.

Given a set of $N = |\mathcal{S}|$ figure-ground segments, the total complexity for inference is $O(Nd^3 + NT + N)$, where $O(Nd^3)$ steps are required to sample up to $N$ tilings [1], with $d = \max_{s_i \in \mathcal{S}} \{|\mathcal{N}^t_{s_i}|\}$, $NT$ is the complexity for inference with TRW-S (with complexity, say, $T$) for each computed tiling, and $N$ steps are done to select the highest scoring labeling. For $|\mathcal{S}| = 200$ the joint inference over labelings and tilings takes under 10 seconds per image in our implementation and produces a set of plausible segmentation and labeling hypotheses which are also useful for learning, described next.

## 4    Incremental Saddle Point Learning

Fundamental to maximum likelihood learning is a tractable, yet stable and sufficiently accurate estimate of the partition function in (7). The number of terms in $Z_\theta(\mathcal{I})$ is $|\mathcal{L}(\mathcal{I})|$ (1), and is exponential both in the number of figure-ground segments and in the number of labels. As reviewed in sec. 3, we approximate the tilings distribution of an image by a number of configurations bounded above by the number of figure-ground segments. This replaces one exponential set of terms in the partition function in (2) (the sum over tilings) with a set of size at most $|\mathcal{S}|$.

In turn, each tiling can be labeled in exponentially many ways—the second sum in the partition function in (2), running over all labelings of a tiling. One possibility to deal with this exponential sum for models with loopy dependencies would be Pseudo-Marginal Approximation (PMA) which estimates $Z_\theta(\mathcal{I})$ using loopy BP and computes gradients as expectations from estimated marginals. Kumar *et al.* [22] found this approximation to perform best for learning conditional random fields for pixel labeling. However it requires inference over all tilings at every optimization iteration. With 484 iterations required for convergence on the VOC dataset, this strategy took in our case 140 times longer than the learning strategy based on incremental saddle-point approximations presented (below), which requires 1.3 hours for learning. Run for the same time, the PMA did not produce satisfactory results in our model (sec. 5).

Another possibility would be to approximate the exponential sum over labels with its largest term, obtained at the most probable configuration (the saddle-point approximation). However, this approach tends to behave erratically as a result of flips within the MAP configurations used to approximate the partition function (sec. 5).

To ensure stability and learning accuracy, we use an incremental saddle point approximation to the partition function. This is obtained by accumulating new incorrect ('offending') labelings rated as the most probable by our current model, in each learning iteration ($\mathcal{L}^j(\mathcal{I})$ denotes the set over which the partition function for image $\mathcal{I}$ is computed in learning iteration $j$):

$$\mathcal{L}^{j+1}(\mathcal{I}) = \mathcal{L}^j(\mathcal{I}) \cup \{\hat{l}, t\} \quad \text{with } (\hat{l}, t) = \underset{l(t),t}{\operatorname{argmax}} \mathcal{F}_\theta(l(t), t, \mathcal{I}) \qquad (8)$$

and $\hat{l} \neq l^\mathcal{I}$ with $l^\mathcal{I}$ the ground truth labeling for image $\mathcal{I}$. We set $\mathcal{L}^0(\mathcal{I}) = \emptyset$. The configurations in $\mathcal{L}^j$ are also used to compute the (analytic) gradient and we use quasi-Newton methods to optimize (7). As learning progresses, new labelings are added to the partition function estimate and this becomes more accurate.

Our learning procedure stops either when (1) all label configurations have been incrementally generated, case when the exact value of the partition function and unbiased estimates for parameters are obtained, or (2) when a subset of the configuration space has been considered in the partition function approximation and no new 'offending' configurations outside this set have been generated during the previous learning (and inference) iteration. In this case a biased estimate is obtained. This is to some extent inevitable for learning models with loopy dependencies and exponential state spaces. In practice, for all datasets we worked on, the learning algorithm converged in 10-25 iterations. In experiments (sec. 5), we show that learning is significantly more stable over standard saddle-point approximations.

## 5   Experiments

We evaluate the quality of semantic segmentation produced by our models in two different datasets: the Stanford Background Dataset [2], and the VOC2010 Pascal Segmentation Challenge [23].

The Stanford Background Dataset contains 715 images and comprises two domains of annotation: semantic classes and geometric classes. The task is to label each pixel in every image with both types of properties. The dataset also contains mid-level segmentation annotations for individual objects, which we use to initially learn the parameters of the segmentation model (see sec. 3 and [1]). Evaluation in this dataset is performed using cross-validation over 5 folds, as in [2]. The evaluation criterion is the mean pixel (labeling) accuracy.

The VOC2010 dataset is accepted as currently one of the most challenging object-class segmentation benchmarks. This dataset also has annotation for individual objects, which we use to learn mid-level segmentation parameters ($\beta$). Unlike Stanford, where all pixels are annotated, on VOC only objects from the 20 classes have ground truth labels. The evaluation criterion is the VOC score: the average per-class overlap between pixels labeled in each class and the respective ground truth annotation[3].

**Quality of segments and tilings:** We generate a bag of figure-ground segments for each image using the publicly available CPMC code [3]. CPMC is an algorithm that generates a large pool (or bag) of figure-ground segmentations, scores them using mid-level properties, and returns the

|                        | Max. pixel accuracy |
|------------------------|---------------------|
| Stanford Geometry      | 93.3                |
| Stanford Semantics     | 85.6                |
|                        | Max. VOC score      |
| VOC2010 Object Classes | 77.9                |

| Method            | Semantic | Geometry |
|-------------------|----------|----------|
| JSL               | 75.6     | 88.8     |
| Gould *et al.* [2]| 76.4     | 91.0     |

Table 1: *Left:* Study of maximum achievable labeling accuracy for our tiling set, for Stanford and VOC2010. The study uses our tiling closest to the segmentation ground truth and assigns 'perfect' pixel labels to it based on that ground truth. In contrast, the best labeling accuracy we obtain automatically is 88.8 for Stanford Geometry, 75.6 for Stanford Semantic, and 41.7 for VOC2010. This shows that potential bottlenecks in reaching the maximum values have to do more with training (ranking) and labeling, rather than the spatial segment layouts and the tiling configurations produced. The average number of segments per tiling are 6.6 on Stanford and 7.9 on VOC. *Right:* Mean pixel accuracies on the Stanford Labeling Dataset. We obtain results comparable to the state-of-the-art in a challenging full-image labeling problem. The results are significant, considering that we use tilings (image segmentations) made on average of 6.6 segments per image. The same method is also competitive in object segmentation datasets such as the VOC2010, where the object granularity is much higher and regions with large spatial support are decisive for effective recognition (table 2).

top $k$ ranked. The online version contains pre-trained models on VOC, but these tend to discard background regions, since VOC has none. For the Stanford experiments, we retrain the CPMC segment ranker using Stanford's segment layout annotations. We generated segment bags having up to 200 segments on the Stanford dataset, and up to 100 segments on the VOC dataset. We model and sample tilings using the methodology described in [1] (see also (5) and sec. 3).

Table 1, left) gives labeling performance upper-bounds on the two datasets for the figure-ground segments and tilings produced. It can be seen that the upper bounds are high for both problems, hence the quality of segments and tilings do not currently limit the final labeling performance, compared to the current state-of-the-art. For further detail on the figure-ground segment pool quality (CPMC) and their assembly into complete image interpretations (FGtiling), we refer to [3, 1].

**Labeling performance:** The tiling component of our model (5) has 41 unary and 31 pairwise parameters ($\beta$) in VOC2010, and 40 unary and 74 parameters ($\beta$) in Stanford. Detail for these features is given in [1]. We will discuss only the features used by the labeling component of the model (4) in this section.

In both VOC2010 and Stanford we use two meta-features for the unary, category-dependent terms. One type of meta-feature is produced as the output of regressors trained (on specific image features described next) to predict overlap of input segments to putative categories. There is one such meta-feature (1 regressor) for each category. A second type of meta-feature is obtained from an object detector [24] to which a particular segment is presented. These detectors operate on bounding boxes, so we determine segment class scores as those of the bounding box overlapping most with the bounding box enclosing each segment.

Since the target semantic concepts of the Stanford and VOC2010 datasets are widely different, we use label-dependent unary terms based on different features. In both cases we use pairwise features connecting all segments ($\mathcal{N}_s^l$ encodes full connectivity), among those belonging to a same tiling. As pairwise features for $\Psi^l$ we use simply a square matrix with all values set to 1, as in [5]. In this way, the model can learn to avoid unlikely patterns of label co-occurrence.

On the Stanford Background Dataset, we train two types of unary meta-features for each class, for semantic and geometric classes. The first unary meta-feature is the output of a regressor trained with the publicly available features from Hoiem *et al.* [7], and the second one uses the features of Gould *et al.* [25]. Each of the feature vectors is transformed using a randomized feature map that approximates the Gaussian-RBF kernel [26, 27]. Using this methodology we can work with linear models in the randomized feature map, yet exploit non-linear kernel embeddings. Summarizing, for Stanford geometry, we have 12 parameters, $\alpha$ (9 unary parameters from 3 classes, each with 2 meta-features and bias and 3 pairwise parameters), whereas for Stanford semantic labels we have 52 parameters, $\alpha$ (24 unary from 8 classes, each with 2 meta-features and bias, and 28 pair-wise, the upper triangle of an 8x8 matrix).

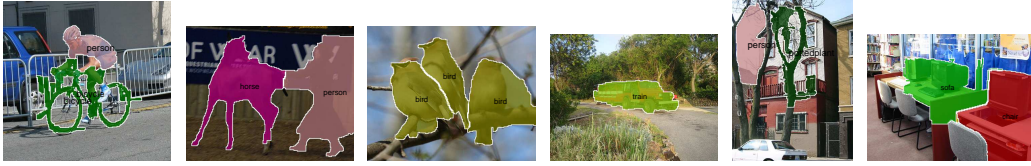

Figure 2: (Best viewed in color) Semantic segmentation results of our method on images from the VOC2010 test set: first three images where the algorithm performs satisfactorily, whereas the last three examples where the algorithm works less well. Notice that identifying multiple objects from the same class is possible in this framework.

In the Stanford dataset, background regions such as grass and sky are shapeless and often locally discriminative. In such cases methods relying on pixel-level descriptors usually obtain good results (e.g. see baseline in [2]). In turn, outdoor datasets containing stuff are challenging for a method like ours that relies on segmentations (tilings) which have an average of 6.6 segments per image (table 1, left). The results we obtain are comparable to Gould *et al.* [2], as visible in table 1, right. The evaluation criterion is the same for both methods: the mean pixel accuracy.

On the VOC2010 dataset, performance is evaluated using the *VOC score*, the average of per-class overlap between pixels labeled in each class and the respective ground truth class. We used two different unary meta-features as well. The first is the output of SVM regressors trained as in [28] using their publicly available features [3]. These regressors predict class scores directly on segments, based on several features: bag of words of gray-level SIFT [29] and color SIFT [30] defined on the foreground and background of each individual segment, and three pyramid HOGs with different parameters. Multiple chi-square kernels $K(x, y) = \exp(-\gamma\chi^2(x, y))$ are combined as in [28]. As a second unary meta-feature we use the outputs of deformable part model detectors [24]. Summarizing, we have 63 category-dependent unary parameters, $\alpha$ (21 classes, each having 2 meta-features and bias), and 210 category-dependent pairwise parameters $\alpha$ (upper triangle of 21x21 matrix). The results, which match and slightly improve the recent winners in the 2010 VOC challenge, are reported in table 2. In particular, our method produces the highest VOC score average over all classes, and also scores first on 9 individual classes. The images in fig. 2 show that our algorithm produces correct labelings. Notice that often the boundaries produced by tilings align with the boundaries of individual objects, even when there are multiple such nearby objects from the same class.

**Impact of different segmentation and labeling methods:** We also evaluate the inference method of [4] (using the code provided by the authors), on the VOC 2010 dataset, and the same input segments and potentials as for JSL. The inference time of the C++ implementation of [4] is comparable with our MATLAB implementations of FGtiling and JSL. The score obtained by [4] on our model is 31.89%, 2.8% higher than the score obtained by the authors using piece-wise training and a dif-

| Classes | JSL | CHD | BSS | Classes | JSL | CHD | BSS | Classes | JSL | CHD | BSS |
|---|---|---|---|---|---|---|---|---|---|---|---|
| Background | 83.4 | 81.1 | **84.2** | Cat | **44.6** | 31.9 | 42.6 | PottedPlant | 19.3 | 30.1 | **36.8** |
| Aeroplane | 51.6 | **58.3** | 52.5 | Chair | **10.6** | 9.1 | 9.0 | Sheep | 45.0 | 36.8 | **50.3** |
| Bicycle | 25.1 | 23.1 | **27.4** | Cow | **41.2** | 36.8 | 32.9 | Sofa | **24.4** | 19.4 | 21.9 |
| Bird | **52.4** | 39.0 | 32.3 | DiningTable | **29.9** | 24.6 | 25.2 | Train | 37.2 | 44.1 | 35.2 |
| Boat | 35.6 | **37.8** | 34.5 | Dog | 25.5 | 29.4 | 27.1 | Tv/Monitor | 43.3 | 35.9 | 40.9 |
| Bottle | **49.6** | 36.4 | 47.4 | Horse | **49.8** | 37.5 | 32.4 | | | | |
| Bus | **66.7** | 63.2 | 60.6 | Motorbike | 47.9 | **60.6** | 47.1 | | | | |
| Car | 55.6 | **62.4** | 54.8 | Person | 37.2 | **44.9** | 38.3 | *Average* | **41.7** | 40.1 | 39.7 |

Table 2: Per class results and averages obtained by our method (JSL) as well as top-scoring methods in the VOC2010 segmentation challenge (CHD: CVC-HARMONY-DET [15], BSS: BONN-SVR-SEGM [28]). Compared to other VOC2010 participants, the proposed method obtains better scores in 9 out of 21 classes, and has superior class average, the standard measure used for ranking. Top scores for each class are marked in bold. Results for other methods can be found in [23]. Note that both JSL (the meta-features) and CHD are trained with the additional bounding box data and images from the training set for object detection. Using this additional training data the class average obtained by BSS is 43.8 [28].

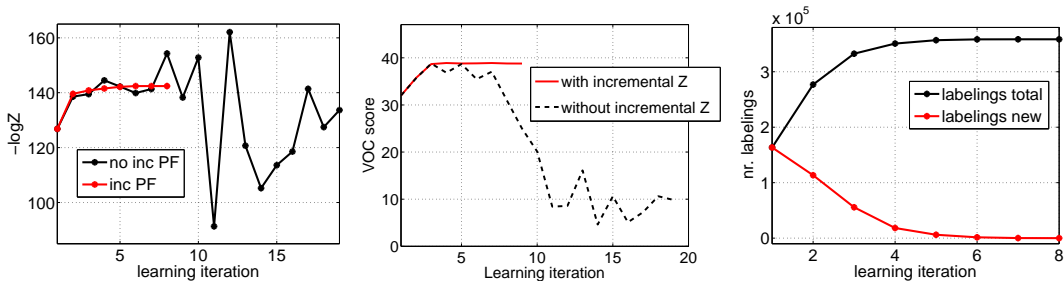

Figure 3: *Left:* The negative $\log(Z)$ at the end of each iteration, for standard (non-incremental) and incremental saddle-point approximations to partition function. Without the stable and more accurate incremental saddle-point approximation to the partition function, the algorithm cannot successfully learn. Results are obtained by training on VOC2010's 'trainval' (train+validation) dataset. *Center:* VOC2010 labeling score as a function of the learning iteration (training on VOC2010's 'trainval'). *Right:* Number of new labeling configurations added to the partition function expansion as learning proceeds for VOC2010. Most configurations are added in the first few iterations.

ferent pool of segments [23], but 9.8% lower than the score of JSL. This suggests that a layered strategy based on selecting a compact set of representative segmentations, followed by labeling is more accurate than sequentially searching for segments and their labels.

In practice, the proposed JSL framework does not depend on FGtiling/CPMC to provide segmentations. Instead, we can use any segmentation method. We have tested the JSL framework (learning and inference) on the Stanford dataset, using segmentations produced by the Ultrametric Contour Map (UCM) hierarchical segmentation method [9]. To obtain a similar number of segments as for CPMC (200 per image), we have selected only the segmentation levels above 20. The features and parameters where computed exactly as before. The bag of segments for each image was derived from the UCM segmentations, and the segmentations where taken as tiling configurations for the corresponding image. In this case, the scores are 76.8 and 88.2 for the semantic and geometric classes, respectively, showing the robustness of JSL to different input segmentations (see also table 1, right).

**Learning performance:** In all our learning experiments, the model parameters have been initialized to the null vector, before learning proceeds, except for the $\alpha$ corresponding to the unary terms in $\mathcal{F}^l_\alpha$ which where set to one. Figure 3, left and center, shows comparisons of learning with and without the incremental saddle point approximation to the partition function, for the VOC 2010 dataset. Without accumulating labelings incrementally, the learning algorithm exhibits erratic behavior and overfits—the relatively small number of labelings used to estimate the partition function produce very different results between consecutive iterations. Figure 3, right, shows the number of total and new labelings added at each learning iteration.

Learning the parameters on VOC 2010 using PMA has taken 180 hours and produced a VOC score of 41.3%. Stopping the learning with PMA after 2 hours (slightly above the 1.3 hrs required by the incremental saddle point approximation) results in a VOC score of 3.87%.

## 6   Conclusion

We have presented a joint image segmentation and labeling model (JSL) which, given a bag of figure-ground image segment hypotheses, constructs a joint probability distribution over *both* the compatible image interpretations assembled from those segments, *and* over their labeling. The process can be interpreted as first sampling maximal cliques from a graph connecting all segments that do not spatially overlap, followed by sampling labels for those segments, conditioned on the choice of their particular tiling. We propose a joint learning procedure based on Maximum Likelihood where the partition function over tilings and labelings is increasingly more accurately approximated during training, by including incorrect configurations that the model rates probable. This ensures that mistakes are not carried on uncorrected in future training iterations, and produces stable and accurate learning schedules. We show that models can be learned efficiently and match the state of the art in the Stanford dataset, as well as VOC2010 where 41.7% accuracy on the test set is achieved.

## Footnotes

[1]Some of the figure-ground segments in $\mathcal{S}(I)$ can spatially overlap.

[2]We call a segmentation assembled from non-overlapping figure-ground segments a tiling, and the tiling together with the set of corresponding labels for its segments a labeling (rather than a labeled tiling).

[3]The overlap measure of two segments is $O(s, s^g) = \frac{|s \cap s^g|}{|s \cup s^g|}$ [23].

# References

[1] A. Ion, J. Carreira, and C. Sminchisescu. Image segmentation by figure-ground composition into maximal cliques. In *ICCV*, November 2011.

[2] S. Gould, R. Fulton, and D. Koller. Decomposing a scene into geometric and semantically consistent regions. In *ICCV*, September 2009.

[3] J. Carreira and C. Sminchisescu. Constrained parametric min-cuts for automatic object segmentation. In *CVPR*, June 2010.

[4] M. P. Kumar and D. Koller. Efficiently selecting regions for scene understanding. In *CVPR*, 2010.

[5] S. Nowozin, P.V. Gehler, and C.H. Lampert. On parameter learning in crf-based approaches to object class image segmentation. In *ECCV*, 2010.

[6] L. Ladicky, C. Russell, P. Kohli, and P. H. S. Torr. Associative hierarchical crfs for object class image segmentation. In *ICCV*, 2009.

[7] D. Hoiem, A. Efros, and M. Hebert. Recovering surface layout from an image. *IJCV*, 75(1), 2007.

[8] K. Barnard, P. Duygulu, D. Forsyth, N. de Freitas, D. M. Blei, and M. Jordan. Matching words and pictures. *JMLR.*, 3:1107–1135, March 2003.

[9] P. Arbelaez, M. Maire, C. Fowlkes, and J. Malik. From contours to regions: An empirical evaluation. In *CVPR*, pages 2294–2301, June 2009.

[10] T. Malisiewicz and A. Efros. Improving spatial support for objects via multiple segmentations. In *BMVC*, 2007.

[11] J. Shotton, J. Winn, C. Rother, and A. Criminisi. Textonboost for image understanding: Multi-class object recognition and segmentation by jointly modeling texture, layout, and context. *IJCV*, 81:2–23, 2009.

[12] X. He, R. S. Zemel, and M. Carreira-Perpinan. Multiscale conditional random fields for image labeling. *CVPR*, 2004.

[13] G. Csurka and F. Perronnin. An efficient approach to semantic segmentation. *IJCV*, pages 1–15, 2010.

[14] B. Fulkerson, A. Vedaldi, and S. Soatto. Class segmentation and object localization with superpixel neighborhoods. In *ICCV*, 2009.

[15] J. M. Gonfaus, X. Boix, J. van de Weijer, A. D. Bagdanov, J Serrat, and J. Gonzalez. Harmony potentials for joint classification and segmentation. In *CVPR*, 2010.

[16] P. Kohli, L. Ladicky, and P.H.S. Torr. Robust higher order potentials for enforcing label consistency. In *CVPR*, 2008.

[17] L. Ladicky, P. Sturgess, K. Alaharia, C. Russel, and P.H.S. Torr. What, where & how many ? combining object detectors and crfs. In *ECCV*, September 2010.

[18] C. Pantofaru, C. Schmid, and M. Hebert. Object recognition by integrating multiple image segmentations. In *ECCV*, 2008.

[19] J.J. Lim, P. Arbelaez, Chunhui Gu, and J. Malik. Context by region ancestry. In *ICCV*, 2009.

[20] Z. Tu, X. Chen, A.L. Yuille, and S.-C. Zhu. Image parsing: unifying segmentation, detection, and recognition. In *ICCV*, 2003.

[21] V. Kolmogorov. Convergent tree-reweighted message passing for energy minimization. *PAMI*, 28(10):1568–1583, 2006.

[22] S. Kumar, J. August, and M. Hebert. Exploiting inference for approximate parameter learning in discriminative fields: An empirical study. In *EMMCVPR*, 2005.

[23] M. Everingham, L. Van Gool, C. K. I. Williams, J. Winn, and A. Zisserman. The PASCAL Visual Object Classes Challenge 2010 (VOC2010) Results. http://www.pascal-network.org/challenges/VOC/.

[24] P. F. Felzenszwalb, R. B. Girshick, D. McAllester, and D. Ramanan. Object detection with discriminatively trained part-based models. *PAMI*, 32(9):1627–1645, 2010.

[25] S. Gould, J. Rodgers, D. Cohen, G. Elidan, and D. Koller. Multi-class segmentation with relative location prior. *IJCV*, 80(3):300–316, 2008.

[26] A. Rahimi and B. Recht. Random features for large-scale kernel machines. In *NIPS*, December 2007.

[27] F. Li, C. Ionescu, and C. Sminchisescu. Random Fourier approximations for skewed multiplicative histogram kernels. In *DAGM*, September 2010.

[28] F. Li, J. Carreira, and C. Sminchisescu. Object recognition by sequential figure-ground ranking. *IJCV*, 2012.

[29] D. G. Lowe. Distinctive image features from scale-invariant keypoints. *IJCV*, 60(2):91–110, 2004.

[30] K. E. A. van de Sande, T. Gevers, and C. G. M. Snoek. Evaluating color descriptors for object and scene recognition. *PAMI*, 32(9):1582–1596, 2010.

